# Stable Dynamic Parameter Adaptation

**Stefan M. Rüger**
Fachbereich Informatik, Technische Universität Berlin
Sekr. FR 5-9, Franklinstr. 28/29
10 587 Berlin, Germany
async@cs.tu-berlin.de

## Abstract

A stability criterion for dynamic parameter adaptation is given. In the case of the learning rate of backpropagation, a class of stable algorithms is presented and studied, including a convergence proof.

## 1 INTRODUCTION

All but a few learning algorithms employ one or more parameters that control the quality of learning. Backpropagation has its learning rate and momentum parameter; Boltzmann learning uses a simulated annealing schedule; Kohonen learning a learning rate and a decay parameter; genetic algorithms probabilities, etc. The investigator always has to set the parameters to specific values when trying to solve a certain problem. Traditionally, the metaproblem of adjusting the parameters is solved by relying on a set of well-tested values of other problems or an intensive search for good parameter regions by restarting the experiment with different values. In this situation, a great deal of expertise and/or time for experiment design is required (as well as a huge amount of computing time).

### 1.1 DYNAMIC PARAMETER ADAPTATION

In order to achieve dynamic parameter adaptation, it is necessary to modify the learning algorithm under consideration: evaluate the performance of the parameters in use from time to time, compare them with the performance of nearby values, and (if necessary) change the parameter setting on the fly. This requires that there exist a measure of the quality of a parameter setting, called performance, with the following properties: the performance depends continuously on the parameter set under consideration, and it is possible to evaluate the performance locally, i.e., at a certain point within an inner loop of the algorithm (as opposed to once only at the end of the algorithm). This is what *dynamic parameter adaptation* is all about.

Dynamic parameter adaptation has several virtues. It is automatic; and there is no
need for an extra schedule to find what parameters suit the problem best. When
the notion of what the good values of a parameter set are changes during learning,
dynamic parameter adaptation keeps track of these changes.

## 1.2  EXAMPLE: LEARNING RATE OF BACKPROPAGATION

Backpropagation is an algorithm that implements gradient descent in an error
function $E\colon \mathbb{R}^n \to \mathbb{R}$. Given $w^o \in \mathbb{R}^n$ and a fixed $\eta > 0$, the iteration rule is
$w^{t+1} = w^t - \eta \nabla E(w^t)$. The learning rate $\eta$ is a local parameter in the sense that
at different stages of the algorithm different learning rates would be optimal. This
property and the following theorem make $\eta$ especially interesting.

*Trade-off theorem for backpropagation. Let $E\colon \mathbb{R}^n \to \mathbb{R}$ be the error function of
a neural net with a regular minimum at $w^* \in \mathbb{R}^n$, i.e., $E$ is expansible into a
Taylor series about $w^*$ with vanishing gradient $\nabla E(w^*)$ and positive definite Hessian
matrix $H(w^*)$. Let $\lambda$ denote the largest eigenvalue of $H(w^*)$. Then, in general,
backpropagation with a fixed learning rate $\eta > 2/\lambda$ cannot converge to $w^*$.*

*Proof.* Let $U$ be an orthogonal matrix that diagonalizes $H(w^*)$, i.e., $D :=
U^T H(w^*)U$ is diagonal. Using the coordinate transformation $x = U^T(w - w^*)$
and Taylor expansion, $E(w) - E(w^*)$ can be approximated by $F(x) := x^T Dx/2$.
Since gradient descent does not refer to the coordinate system, the asymptotic be-
havior of backpropagation for $E$ near $w^*$ is the same as for $F$ near 0. In the latter
case, backpropagation calculates the weight components $x_i^t = x_i^o(1 - D_{ii}\eta)^t$ at time
step $t$. The diagonal elements $D_{ii}$ are the eigenvalues of $H(w^*)$; convergence for all
geometric sequences $t \mapsto x_i^t$ thus requires $\eta < 2/\lambda$. ∎

The trade-off theorem states that, given $\eta$, a large class of minima cannot be found,
namely, those whose largest eigenvalue of the corresponding Hessian matrix is larger
than $2/\eta$. Fewer minima might be overlooked by using a smaller $\eta$, but then the
algorithm becomes intolerably slow. Dynamic learning-rate adaptation is urgently
needed for backpropagation!

## 2  STABLE DYNAMIC PARAMETER ADAPTATION

Transforming the equation for gradient descent, $w^{t+1} = w^t - \eta \nabla E(w^t)$, into a
differential equation, one arrives at $\partial w^t/\partial t = -\eta \nabla E(w^t)$. Gradient descent with
constant step size $\eta$ can then be viewed as Euler's method for solving the differential
equation. One serious drawback of Euler's method is that it is unstable: each finite
step leaves the trajectory of a solution without trying to get back to it. Virtually
any other differential-equation solver surpasses Euler's method, and there are even
some featuring dynamic parameter adaptation [5].

However, in the context of function minimization, this notion of stability ("do not
drift away too far from a trajectory") would appear to be too strong. Indeed,
differential-equation solvers put much effort into a good estimation of points that
are as close as possible to the trajectory under consideration. What is really needed
for minimization is *asymptotic stability*: ensuring that the performance of the pa-
rameter set does not decrease at the end of learning. This weaker stability criterion
allows for greedy steps in the initial phase of learning.

There are several successful examples of dynamic learning-rate adaptation for back-
propagation: Newton and quasi-Newton methods [2] as an adaptive $\eta$-tensor; indi-
vidual learning rates for the weights [3, 8]; conjugate gradient as a one-dimensional
$\eta$-estimation [4]; or straightforward $\eta$-adaptation [1, 7].

A particularly good example of dynamic parameter adaptation was proposed by Salomon [6, 7]: let $\zeta > 1$; at every step $t$ of the backpropagation algorithm test two values for $\eta$, a somewhat smaller one, $\eta_t/\zeta$, and a somewhat larger one, $\eta_t\zeta$; use as $\eta_{t+1}$ the value with the better performance, i.e., the smaller error:

$$\eta_{t+1} = \begin{cases} \eta_t/\zeta & \text{if } E(w^t - \eta_t/\zeta \cdot \nabla E(w^t)) \leq E(w^t - \eta_t\zeta \cdot \nabla E(w^t)) \\ \eta_t\zeta & \text{otherwise} \end{cases}$$

The setting of the new parameter $\zeta$ proves to be uncritical (all values work, especially sensible ones being those between 1.2 and 2.1). This method outperforms many other gradient-based algorithms, but it is nonetheless unstable.

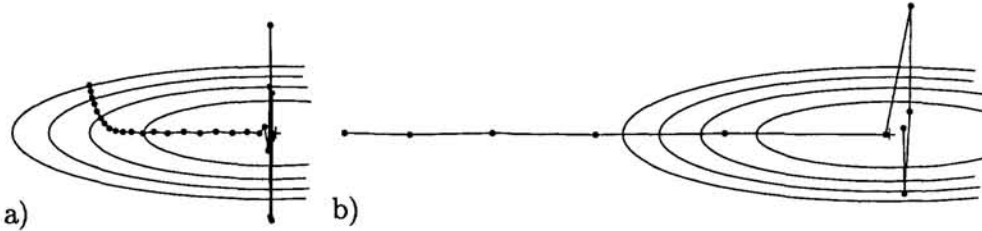

Figure 1: Unstable Parameter Adaptation

The problem arises from a rapidly changing length and direction of the gradient, which can result in a huge leap away from a minimum, although the latter may have been almost reached. Figure 1a shows the niveau lines of a simple quadratic error function $E: \mathbb{R}^2 \to \mathbb{R}$ along with the weight vectors $w^o, w^1, \ldots$ (bold dots) resulting from the above algorithm. This effect was probably the reason why Salomon suggested using the normalized gradient instead of the gradient, thus getting rid of the changes in the length of the gradient. Although this works much better, Figure 1b shows the instability of this algorithm due to the change in the gradient's direction.

There is enough evidence that these algorithms converge for a purely quadratic error function [6, 7]. Why bother with stability? One would like to prove that an algorithm asymptotically finds the minimum, rather than occasionally leaping far away from it and thus leaving the region where the quadratic Hessian term of a globally nonquadratic error function dominates.

# 3   A CLASS OF STABLE ALGORITHMS

In this section, a class of algorithms is derived from the above ones by adding stability. This class provides not only a proof of asymptotic convergence, but also a significant improvement in speed.

*Let $E: \mathbb{R}^n \to \mathbb{R}$ be an error function of a neural net with random weight vector $w^o \in \mathbb{R}^n$. Let $\zeta > 1$, $\eta_o > 0$, $0 < c \leq 1$, and $0 < a \leq 1 \leq b$. At step $t$ of the algorithm, choose a vector $g^t$ restricted only by the conditions $g^t \nabla E(w^t)/|g^t|\|\nabla E_{w^t}| \geq c$ and that it either holds for all $t$ that $1/|g^t| \in [a, b]$ or that it holds for all $t$ that $|\nabla E(w^t)|/|g^t| \in [a, b]$, i.e., the vectors $g^t$ have a minimal positive projection onto the gradient and either have a uniformly bounded length or are uniformly bounded by the length of the gradient. Note that this is always possible by choosing $g^t$ as the gradient or the normalized gradient.*

*Let $e: \eta \mapsto E(w^t - \eta g^t)$ denote a one-dimensional error function given by $E$, $w^t$ and $g^t$. Repeat (until the gradient vanishes or an upper limit of $t$ or a lower limit $E_{\min}$*

*of $E$ is reached) the iteration $w^{t+1} = w^t - \eta_{t+1} g^t$ with*

$$\eta_{t+1} = \begin{cases} \eta^* := \dfrac{\eta_t \zeta/2}{1 + \dfrac{e(\eta_t \zeta) - e(0)}{\eta_t \zeta g^t \nabla E(w^t)}} & \text{if } e(0) < e(\eta_t \zeta) \\[2em] \eta_t/\zeta & \text{if } e(\eta_t/\zeta) \leq e(\eta_t \zeta) \leq e(0) \\ \eta_t \zeta & \text{otherwise.} \end{cases} \tag{1}$$

The first case for $\eta_{t+1}$ is a stabilizing term $\eta^*$, which definitely decreases the error when the error surface is quadratic, i.e., near a minimum. $\eta^*$ is put into effect when the error $e(\eta_t \zeta)$, which would occur in the next step if $\eta_{t+1} = \eta_t \zeta$ was chosen, exceeds the error $e(0)$ produced by the present weight vector $w^t$. By construction, $\eta^*$ results in a value less than $\eta_t \zeta/2$ if $e(\eta_t \zeta) > e(0)$; hence, given $\zeta < 2$, the learning rate is decreased as expected, no matter what $E$ looks like. Typically, (if the values for $\zeta$ are not extremely high) the other two cases apply, where $\eta_t \zeta$ and $\eta_t/\zeta$ compete for a lower error.

Note that, instead of gradient descent, this class of algorithms proposes a "$g^t$ descent," and the vectors $g^t$ may differ from the gradient. A particular algorithm is given by a specification of how to choose $g^t$.

## 4   PROOF OF ASYMPTOTIC CONVERGENCE

**Asymptotic convergence.** *Let $E: w \mapsto \sum_{i=1}^n \lambda_i w_i^2/2$ with $\lambda_i > 0$. For all $\zeta > 1$, $0 < c \leq 1$, $0 < a \leq 1 \leq b$, $\eta_o > 0$, and $w^o \in \mathbb{R}^n$, every algorithm from Section 3 produces a sequence $t \mapsto w^t$ that converges to the minimum 0 of $E$ with an at least exponential decay of $t \mapsto E(w^t)$.*

*Proof.* This statement follows if a constant $q < 1$ exists with $E(w^{t+1}) \leq qE(w^t)$ for all $t$. Then, $\lim_{t \to \infty} w^t = 0$, since $w \mapsto \sqrt{E(w)}$ is a norm in $\mathbb{R}^n$.

Fix a $w^t$, $\eta_t$, and a $g^t$ according to the premise. Since $E$ is a positive definite quadratic form, $e: \eta \mapsto E(w^t - \eta g^t)$ is a one-dimensional quadratic function with a minimum at, say, $\eta^*$. Note that $e(0) = E(w^t)$ and $e(\eta_{t+1}) = E(w^{t+1})$. $e$ is completely determined by $e(0)$, $e'(0) = -g^t \nabla E(w^t)$, $\eta_t \zeta$, and $e(\eta_t \zeta)$. Omitting the algebra, it follows that $\eta^*$ can be identified with the stabilizing term of (1).

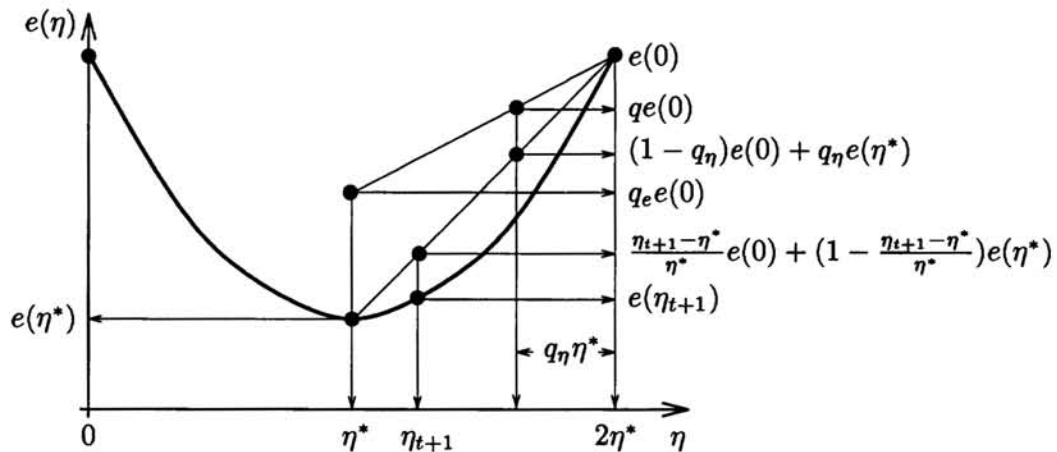

Figure 2: Steps in Estimating a Bound $q$ for the Improvement of $E$.

If $e(\eta_t \zeta) > e(0)$, by (1) $\eta_{t+1}$ will be set to $\eta^*$; hence, $w^{t+1}$ has the smallest possible error $e(\eta^*)$ along the line given by $g^t$. Otherwise, the three values $0$, $\eta_t/\zeta$, and $\eta_t \zeta$ cannot have the same error $e$, as $e$ is quadratic; $e(\eta_t \zeta)$ or $e(\eta_t/\zeta)$ must be less than $e(0)$, and the argument with the better performance is used as $\eta_{t+1}$. The sequence $t \mapsto E(w^t)$ is strictly decreasing; hence, a $q \leq 1$ exists. The rest of the proof shows the existence of a $q < 1$.

Assume there are two constants $0 < q_e, q_\eta < 1$ with

$$\eta_{t+1}/\eta^* \quad \in \quad [q_\eta, 2 - q_\eta] \tag{2}$$

$$e(\eta^*) \quad \leq \quad q_e e(0). \tag{3}$$

Let $\eta_{t+1} \geq \eta^*$; using first the convexity of $e$, then (2), and (3), one obtains

$$
\begin{aligned}
e(\eta_{t+1}) &= e\Big(\frac{\eta_{t+1} - \eta^*}{\eta^*} 2\eta^* + \big(1 - \frac{\eta_{t+1} - \eta^*}{\eta^*}\big)\eta^*\Big) \\
&\leq \frac{\eta_{t+1} - \eta^*}{\eta^*} e(0) + \big(1 - \frac{\eta_{t+1} - \eta^*}{\eta^*}\big)e(\eta^*) \\
&\leq (1 - q_\eta)e(0) + q_\eta e(\eta^*) \\
&\leq (1 - q_\eta(1 - q_e))e(0).
\end{aligned}
$$

Figure 2 shows how the estimations work. The symmetric case $0 < \eta_{t+1} \leq \eta^*$ has the same result $E(w^{t+1}) \leq qE(w^t)$ with $q := 1 - q_\eta(1 - q_e) < 1$.

Let $\lambda^< := \min\{\lambda_i\}$ and $\lambda^> := \max\{\lambda_i\}$. A straightforward estimation for $q_e$ yields

$$q_e := 1 - c^2 \frac{\lambda^<}{\lambda^>} < 1.$$

Note that $\eta^*$ depends on $w^t$ and $g^t$. A careful analysis of the recursive dependence of $\eta^{t+1}/\eta^*(w^t, g^t)$ on $\eta^t/\eta^*(w^{t-1}, g^{t-1})$ uncovers an estimation

$$q_\eta := \min\Big\{ \frac{2}{\zeta^2 + 1}, \frac{2\zeta}{\zeta^2 + 1} \frac{ca}{b} \Big(\frac{\lambda^<}{\lambda^>}\Big)^{3/2}, \frac{\eta_o \zeta \lambda^<}{b \max\{1, \sqrt{2\lambda^> E(w^o)}\}} \Big\} > 0. \qquad \blacksquare$$

# 5  NON-GRADIENT DIRECTIONS CAN IMPROVE CONVERGENCE

It is well known that the sign-changed gradient of a function is not necessarily the best direction to look for a minimum. The momentum term of a modified back-propagation version uses old gradient directions; Newton or quasi-Newton methods explicitly or implicitly exploit second-order derivatives for a change of direction; another choice of direction is given by conjugate gradient methods [5].

The algorithms from Section 3 allow almost any direction, as long as it is not nearly perpendicular to the gradient. Since they estimate a good step size, these algorithms can be regarded as a sort of "trial-and-error" line search without bothering to find an exact minimum in the given direction, but utilizing any progress made so far.

One could incorporate the Polak-Ribiere rule, $d^{t+1} = \nabla E(w^{t+1}) + \alpha\beta d^t$, for conjugate directions with $d^o = \nabla E(w^o), \alpha = 1$, and

$$\beta = \frac{(\nabla E(w^{t+1}) - \nabla E(w^t))\nabla E(w^{t+1})}{(\nabla E(w^t))^2}$$

to propose vectors $g^t := d^t/|d^t|$ for an explicit algorithm from Section 3. As in the conjugate gradient method, one should reset the direction $d^t$ after each $n$ (the number of weights) updates to the gradient direction. Another reason for resetting the direction arises when $g^t$ does not have the minimal positive projection $c$ onto the normalized gradient.

$\alpha = 0$ sets the descent direction $g^t$ to the normalized gradient $\nabla E(w^t)/|\nabla E(w^t)|$; this algorithm proves to exhibit a behavior very similar to Salomon's algorithm with normalized gradients. The difference lies in the occurrence of some stabilization steps from time to time, which, in general, improve the convergence.

Since comparisons of Salomon's algorithm to many other methods have been published [7], this paper confines itself to show that significant improvements are brought about by non-gradient directions, e. g., by Polak-Ribiere directions ($\alpha = 1$).

Table 1: Average Learning Time for Some Problems

| PROBLEM | $E_{min}$ | $\alpha = 0$ | $\alpha = 1$ |
|---|---|---|---|
| (a) 3-2-4 regression | $10^0$ | $195 \pm\ 95\%$ | $58 \pm\ 70\%$ |
| (b) 3-2-4 approximation | $10^{-4}$ | $1070 \pm 140\%$ | $189 \pm 115\%$ |
| (c) Pure square ($n = 76$) | $10^{-16}$ | $464 \pm\ 17\%$ | $118 \pm\ 9\%$ |
| (d) Power 1.8 ($n = 76$) | $10^{-4}$ | $486 \pm\ 29\%$ | $84 \pm\ 23\%$ |
| (e) Power 3.8 ($n = 76$) | $10^{-16}$ | $28 \pm\ 10\%$ | $37 \pm\ 14\%$ |
| (f) 8-3-8 encoder | $10^{-4}$ | $1380 \pm\ 60\%$ | $300 \pm\ 60\%$ |

Table 1 shows the average number of epochs of two algorithms for some problems. The average was taken over many initial random weight vectors and over values of $\zeta \in [1.7, 2.1]$; the root mean square error of the averaging process is shown as a percentage. Note that, owing to the two test steps for $\eta_t/\zeta$ and $\eta_t\zeta$, one epoch has an overhead of around 50% compared to a corresponding epoch of backpropagation. $\alpha \neq 0$ helps: it could be chosen by dynamic parameter adaptation.

Problems (a) and (b) represent the approximation of a function known only from some example data. A neural net with 3 input, 2 hidden, and 4 output nodes was used to generate the example data; artificial noise was added for problem (a). The same net with random initial weights was then used to learn an approximation. These problems for feedforward nets are expected to have regular minima.

Problem (c) uses a pure square error function $E: w \mapsto \sum_{i=1}^n i|w_i|^p/2$ with $p = 2$ and $n = 76$. Note that conjugate gradient needs exactly $n$ epochs to arrive at the minimum [5]. However, the few additional epochs that are needed by the $\alpha = 1$ algorithm to reach a fairly small error (here 118 as opposed to 76) must be compared to the overhead of conjugate gradient (one line search per epoch).

Powers other than 2, as used in (d) or (e), work well as long as, say, $p > 1.5$. A power $p < 1$ will (if $n \geq 2$) produce a "trap" for the weight vector at a location near a coordinate axis, where, owing to an infinite gradient component, *no* gradient-based algorithm can escape[1]. Problems are expected even for $p$ near 1: the algorithms of Section 3 exploit the fact that the gradient vanishes at a minimum, which in turn is numerically questionable for a power like 1.1. Typical minima, however, employ powers $2, 4, \ldots$ Even better convergence is expected and found for large powers.

The 8-3-8 encoder (f) was studied, because the error function has global minima at the boundary of the domain (one or more weights with infinite length). These minima, though not covered in Section 4, are quickly found. Indeed, the ability to increase the learning rate geometrically helps these algorithms to approach the boundary in a few steps.

# 6   CONCLUSIONS

It has been shown that implementing asymptotic stability *does* help in the case of the backpropagation learning rate: the theoretical analysis has been simplified, and the speed of convergence has been improved. Moreover, the presented framework allows descent directions to be chosen flexibly, e. g., by the Polak-Ribiere rule. Future work includes studies of how to apply the stability criterion to other parametric learning problems.

## Footnotes

[1]Dynamic parameter adaptation as in (1) can cope with the square-root singularity ($p = 1/2$) in one dimension, because the adaptation rule allows a fast enough decay of the learning rate; the ability to minimize this one-dimensional square-root singularity is somewhat overemphasized in [7].

# References

[1] R. Battiti. Accelerated backpropagation learning: Two optimization methods. *Complex Systems*, 3:331–342, 1989.

[2] S. Becker and Y. le Cun. Improving the convergence of back-propagation learning with second order methods. In D. Touretzky, G. Hinton, and T. Sejnowski, editors, *Proceedings of the 1988 Connectionist Models Summer School*, pages 29–37. Morgan Kaufmann, San Mateo, 1989.

[3] R. Jacobs. Increased rates of convergence through learning rate adaptation. *Neural Networks*, 1:295–307, 1988.

[4] A. Kramer and A. Sangiovanni-Vincentelli. Efficient parallel learning algorithms for neural networks. In D. Touretzky, editor, *Advances in Neural Information Processing Systems 1*, pages 40–48. Morgan Kaufmann, San Mateo, 1989.

[5] W. H. Press, B. P. Flannery, S. A. Teukolsky, and W. T. Vetterling. *Numerical Recipes in C*. Cambridge University Press, 1988.

[6] R. Salomon. *Verbesserung konnektionistischer Lernverfahren, die nach der Gradientenmethode arbeiten*. PhD thesis, TU Berlin, October 1991.

[7] R. Salomon and J. L. van Hemmen. Accelerating backpropagation through dynamic self-adaptation. *Neural Networks*, 1996 (in press).

[8] F. M. Silva and L. B. Almeida. Speeding up backpropagation. In *Proceedings of NSMS - International Symposium on Neural Networks for Sensory and Motor Systems*, Amsterdam, 1990. Elsevier.
